# A Comparison of Discrete-Time Operator Models for Nonlinear System Identification

**Andrew D. Back, Ah Chung Tsoi**
Department of Electrical and Computer Engineering,
University of Queensland
St. Lucia, Qld 4072. Australia.
e-mail: {back,act}@elec.uq.oz.au

## Abstract

We present a unifying view of discrete-time operator models used in the context of finite word length linear signal processing. Comparisons are made between the recently presented gamma operator model, and the delta and rho operator models for performing nonlinear system identification and prediction using neural networks. A new model based on an adaptive bilinear transformation which generalizes all of the above models is presented.

## 1 INTRODUCTION

The shift operator, defined as $qx(t) \stackrel{\triangle}{=} x(t+1)$, is frequently used to provide time-domain signals to neural network models. Using the shift operator, a discrete-time model for system identification or time series prediction problems may be constructed. A common method of developing nonlinear system identification models is to use a neural network architecture as an estimator $\mathcal{F}(Y(t), X(t); \theta)$ of $F(Y(t), X(t))$, where $\theta$ represents the parameter vector of the network. Shift operators at the input of the network provide the regression vectors $Y(t-1) = [y(t-1), ..., y(t-N)]'$, and $X(t) = [x(t), ..., x(t-M)]'$ in a manner analogous to linear filters, where $[\cdot]'$ represents the vector transpose.

It is known that linear models based on the shift operator $q$ suffer problems when used to model lightly-damped-low-frequency (LDLF) systems, with poles near $(1, 0)$ on the unit circle in the complex plane [5]. As the sampling rate increases, coefficient sensitivity and round-off noise become a problem as the difference between successive sampled inputs becomes smaller and smaller.

A method of overcoming this problem is to use an alternative discrete-time operator. Agarwal and Burrus first proposed the use of the *delta* operator in digital filters to replace the shift operator in an attempt to overcome the problems described above [1]. The delta operator is defined as

$$\delta = \frac{q-1}{\Delta} \tag{1}$$

where $\Delta$ is the discrete-time sampling interval. Williamson showed that the delta operator allows better performance in terms of coefficient sensitivity for digital filters derived from the direct form structure [19], and a number of authors have considered using it in linear filtering, estimation and control [5, 7, 8]

More recently, de Vries, Principe at. al. proposed the *gamma* operator [2, 3] as a means of studying neural network models for processing time-varying patterns. This operator is defined by

$$\gamma = \frac{q-(1-c)}{c} \tag{2}$$

It may be observed that it is a generalization of the delta operator with adjustable parameters $c$. An extension to the basic gamma operator introducing complex poles using a second order operator, was given in [18].

This raises the question, is the gamma operator capable of providing better neural network modelling capabilities for LDLF systems ? Further, are there any other operators which may be better than these for nonlinear modelling and prediction using neural networks ?

In the context of robust adaptive control, Palaniswami has introduced the *rho* operator which has shown useful improvements over the performance of the delta operator [9, 10]. The rho operator is defined as

$$\rho = \frac{q-(1-c_1\Delta)}{c_2\Delta} \tag{3}$$

where $c_1, c_2$ are adjustable parameters. The rho operator generalizes the delta and gamma operators. For the case where $c_1\Delta = c_2\Delta = 1$, the rho operator reduces to the usual shift operator . When $c_1 = 0$, and $c_2 = 1$, the rho operator reduces to the delta operator [10]. For $c_1\Delta = c_2\Delta = c$, the rho operator is equivalent to the gamma operator .

One advantage of the rho operator over the delta operator is that it is stably invertible, allowing the derivation of simpler algorithms [9]. The $\rho$ operator can be considered as a stable low pass filter, and parameter estimation using the $\rho$ operator is low frequency biased. For adaptive control systems, this gives robustness advantages for systems with unmodelled high frequency characteristics [9].

By defining the bilinear transformation (BLT) as an operator, it is possible to introduce an operator which generalizes all of the above operators. We can therefore define the *pi* operator as

$$\pi = \frac{2\,(c_1q-c_2)}{\Delta\,(c_3q+c_4)} \tag{4}$$

with the restriction that $c_1c_4 \neq c_2c_3$ (to ensure $\pi$ is not a constant function [14]). The bilinear mapping produced has a pole at $q = -c_4/c_3$. By appropriate setting of the $c_1, c_2, c_3, c_4$ parameters each operator, the pi operator can be reduced to each of the previous operators.

In the work reported here, we consider these alternative discrete-time operators in feed-forward neural network models for system identification tasks. We compare the popular

gamma model [4] with other models based on the shift, delta, rho and pi operators. A framework of models and Gauss-Newton training algorithms is provided, and the models are compared by simulation experiments.

## 2  OPERATOR MODELS FOR NONLINEAR SIGNAL PROCESSING

A model which generalizes the usual discrete-time linear moving average model, ie, a single layer network is given by

$$\hat{y}(t) \ = \ G(\nu, \theta)x(t) \tag{5}$$

$$G(\nu, \theta) \ = \ \sum_{i=0}^{M} b_i \nu^{-i}$$

$$\nu^{-i} \ = \ \begin{cases} q^{-i} & \text{shift operator} \\ \delta^{-i} & \text{delta operator} \\ \gamma^{-i} & \text{gamma operator} \\ \rho^{-i} & \text{rho operator} \\ \pi^{-i} & \text{pi operator} \end{cases} \tag{6}$$

This general class of moving average model can be termed MA($\nu$). We define $u_0(t) \stackrel{\triangle}{=} x(t)$, and $u_i(t) \stackrel{\triangle}{=} \nu^{-1}u_{i-1}(t)$ and hence obtain

$$u_i(t) \ = \ \begin{cases} x(t-i) & \text{shift operator} \\ \Delta u_{i-1}(t-1) + u_i(t-1) & \text{delta operator} \\ cu_{i-1}(t-1) + (1-c)u_i(t-1) & \text{gamma operator} \\ c_2 \Delta u_{i-1}(t-1) + (1-c_1\Delta)u_i(t-1) & \text{rho operator} \\ \frac{\Delta}{2c_1}\left(c_3 u_{i-1}(t) + c_4 u_{i-1}(t-1)\right) - \frac{c_2}{c_1}u_i(t-1) & \text{pi operator} \end{cases} \tag{7}$$

A nonlinear model may be defined using a multilayer perceptron (MLP) with the $\nu$-operator elements at the input stage. The input vector $Z_i^0(t)$ to the network is

$$Z_i^0(t) \ = \ [x_i(t), \nu^{-1}x_i(t), ..., \nu^{-M}x_i(t)]' \tag{8}$$

where $x_i(t)$ is the $i$th input to the system. This model is termed the $\nu$-operator multilayer perceptron or MLP($\nu$) model.

An MLP($\nu$) model having $L$ layers with $N_0, N_1, ..., N_L$ nodes per layer, is defined in the same manner as a usual MLP, with

$$z_k^l(t) \ = \ f\left(\hat{x}_k^l(t)\right) \tag{9}$$

$$\hat{x}_k^l(t) \ = \ \sum_{i=1}^{N_l} w_{ki}^l z_i^{l-1}(t) \tag{10}$$

where each neuron $i$ in layer $l$ has an output of $z_i^l(t)$; a layer consists of $N_l$ neurons ($l = 0$ denotes the input layer, and $l = L$ denotes the output layer, $z_{N_l}^l = 1.0$ may be used for a bias); $f(\cdot)$ is a sigmoid function typically evaluated as $\tanh(\cdot)$, and a synaptic connection between unit $i$ in the previous layer and unit $k$ in the current layer is represented by $w_{ki}^l$. The notation $t$ may be used to represent a discrete time or pattern instance. While the case

we consider employs the $\nu$-operator at the input layer only, it would be feasible to use the operators throughout the network as required.

On-line algorithms to update the operator parameters in the MA($\nu$) model can be found readily. In the case of the MLP($\nu$) model, we approach the problem by backpropagating the error information to the input layer and using this to update the operator coefficients. de Vries and Principe et. al., proposed stochastic gradient descent type algorithms for adjusting the $c$ operator coefficient using a least-squares error criterion [2, 12]. For brevity we omit the updating procedures for the MLP network weights; a variety of methods may be applied (see for example [13, 15]).

We define an instantaneous output error criterion $J(t) = \frac{1}{2}e^2(t)$, where $e(t) = y(t) - \hat{y}(t)$.

Defining $\hat{\theta}$ as the estimated operator parameter vector at time $t$ of the parameter vector $\theta$, we have

$$
\hat{\theta} = \begin{cases} \hat{c} & \text{gamma operator} \\ [\hat{c}_1, \hat{c}_2]' & \text{rho operator} \\ [\hat{c}_1, \hat{c}_2, \hat{c}_3, \hat{c}_4]' & \text{pi operator} \end{cases}
\tag{11}
$$

A first order algorithm to update the coefficients is

$$
\hat{\theta}_i(t+1) = \hat{\theta}_i(t) + \Delta\hat{\theta}_i(t)
\tag{12}
$$

$$
\Delta\hat{\theta}_i(t) = -\eta\nabla_{\theta_i}J(\theta;t)
\tag{13}
$$

where the adjustment in weights is found as

$$
\Delta\hat{\theta}_i(t) = -\eta\frac{\partial J(t)}{\partial\theta_j}
$$

$$
= \eta\sum_{i=1}^{M}\psi_i^{j\,\prime}(t)\delta_j(t)
\tag{14}
$$

where $\delta_j(t)$ is the backpropagated error at the $jth$ node of input layer, and $\psi_i^{j\,\prime}(t)$ is the first order sensitivity vector of the model operator parameters, defined by

$$
\psi_i^j(t) = \begin{cases} \frac{\partial u_i(t)}{\partial c_j} & \text{gamma operator} \\ \left[\frac{\partial u_i(t)}{\partial c_{j1}}, \frac{\partial u_i(t)}{\partial c_{j2}}\right]' & \text{rho operator} \\ \left[\frac{\partial u_i(t)}{\partial c_{j1}}, \frac{\partial u_i(t)}{\partial c_{j2}}, \frac{\partial u_i(t)}{\partial c_{j3}}, \frac{\partial u_i(t)}{\partial c_{j4}}\right]' & \text{pi operator} \end{cases}
\tag{15}
$$

Substituting $u_i(t)$ in from (7), the recursive equations for $\psi_i^j(t)$ (noting that $\psi_i^j(t) = \psi_i'(t)$ $\forall j$) are

$$
\psi_i(t) = u_{i-1}(t-1) - u_i(t-1) + \hat{c}_i\psi_{i-1}(t-1) + (1-\hat{c})\psi_i(t-1) \quad \text{gamma operator}
$$

$$
\psi_i(t) = \begin{bmatrix} \hat{c}_2\Delta\psi_{i-1,1}(t-1) + (1-\hat{c}_1\Delta)\psi_{i,1}(t-1) - \Delta u_i(t-1) \\ \Delta u_{i-1}(t-1) + \hat{c}_2\Delta\psi_{i-1,2}(t-1) + (1-\hat{c}_1\Delta)\psi_{i,2}(t-1) \end{bmatrix} \quad \text{rho operator}
$$

$$
\psi_i(t) = \begin{bmatrix} \frac{\Delta}{2\hat{c}_1}\left(\hat{c}_3\psi_{i-1,1}(t) + \hat{c}_4\psi_{i-1,1}(t-1)\right) + \frac{\hat{c}_2}{\hat{c}_1}\psi_{i,1}(t-1) \\ -\frac{\Delta}{2\hat{c}_1^2}\left(\hat{c}_3u_{i-1}(t) + \hat{c}_4u_{i-1}(t-1)\right) - \frac{\hat{c}_2}{\hat{c}_1^2}u_i(t-1), \\ \frac{\Delta}{2\hat{c}_1}\left(\hat{c}_3\psi_{i-1,2}(t) + \hat{c}_4\psi_{i-1,2}(t-1)\right) + \frac{\hat{c}_2}{\hat{c}_1}\psi_{i,2}(t-1) + \frac{1}{\hat{c}_1}u_i(t-1), \\ \frac{\Delta}{2\hat{c}_1}\left(u_{i-1}(t) + \hat{c}_3\psi_{i-1,3}(t) + \hat{c}_4\psi_{i-1,3}(t-1)\right) + \frac{\hat{c}_2}{\hat{c}_1}\psi_{i,3}(t-1), \\ \frac{\Delta}{2\hat{c}_1}\left(\hat{c}_3\psi_{i-1,4}(t) + u_{i-1}(t-1) + \hat{c}_4\psi_{i-1,4}(t-1)\right) + \frac{\hat{c}_2}{\hat{c}_1}\psi_{i,4}(t-1) \end{bmatrix} \quad \text{pi operator}
$$

for the gamma, rho, and pi operators respectively, and where $\psi_{i,j}(t)$ refers to the $j$th element of the $i$th $\psi$ vector, with $\psi_{i,0}(t) = 0$.

A more powerful updating procedure can be obtained by using the Gauss-Newton method [6]. In this case, we replace (14) with (omitting $i$ subscripts for clarity),

$$\hat{\theta}(t+1) = \hat{\theta}(t) + \gamma(t)R^{-1}(t)\psi(t)\Lambda^{-1}\delta(t) \tag{16}$$

where $\gamma(t)$ is the gain sequence (see [6] for details), $\Lambda^{-1}$ is a weighting matrix which may be replaced by the identity matrix [16], or estimated as [6]

$$\hat{\Lambda}(t) = \hat{\Lambda}(t-1) + \gamma(t)\left(\delta^2(t) - \hat{\Lambda}(t-1)\right) \tag{17}$$

$R(t)$ is an approximate Hessian matrix, defined by

$$R(t+1) = \lambda(t)R(t) + \zeta(t)\psi(t)\psi'(t) \tag{18}$$

where $\lambda(t) = 1 - \zeta(t)$. Efficient computation of $R^{-1}$ may be performed using the matrix inversion lemma [17], factorization methods such as Cholesky decomposition or other fast algorithms. Using the well known matrix inversion lemma [6], we substitute $P(t) = R^{-1}(t)$, where

$$P(t) = \frac{1}{\lambda(t)}P(t) - \frac{\zeta(t)}{\lambda(t)}\left(\frac{P(t)\psi(t)\psi'(t)P(t)}{\lambda(t) + \zeta(t)\psi'(t)P(t)\psi(t)}\right) \tag{19}$$

The initial values of the coefficients are important in determining convergence. Principe et. al. [12] note that setting the coefficients for the gamma operator to unity provided the best approach for certain problems.

## 3 SIMULATION EXAMPLES

We are primarily interested in the differences between the operators themselves for modelling and prediction, and not the associated difficulties of training multilayer perceptrons (recall that our models will only differ at the input layer). For the purposes of a more direct comparison, in this paper we test the models using a single layer network. Hence these linear system examples are used to provide an indication of the operators' performance.

### 3.1 EXPERIMENT 1

The first problem considered is a system identification task arising in the context of high bit rate echo cancellation [5]. In this case, the system is described by

$$H(z) = \frac{0.0254 - 0.0296z^{-1} + 0.00425z^{-2}}{1 - 1.957z^{-1} + 0.957z^{-2}} \tag{20}$$

This system has poles on the real axis at 0.9994, and 0.9577, thus it is an LDLF system. The input signal to the system in each case consisted of uniform white noise with unit variance. A Gauss-Newton algorithm was used to determine all unknown weights. We conducted Monte-Carlo tests using 20 runs of differently seeded training samples each of 2000 points to obtain the results reported. We assessed the performance of the models by using the Signal-to-Noise Ratio (SNR) defined as $10\log(E[d(t)^2]/E[e(t)^2])$, where $E[\cdot]$ is the expectation operator, and $d(t)$ is the desired signal. For each run, we used the last 500 samples to compute a SNR figure.

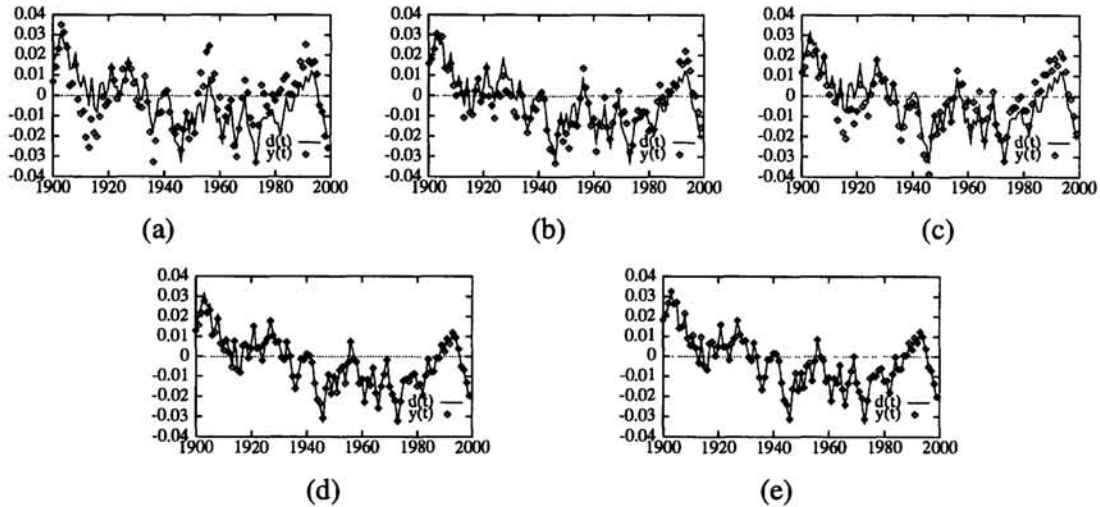

Figure 1: Comparison of typical model output results for Experiment 1 with models based on the following operators: (a) shift, (b) delta (c) gamma, (d) rho, and (e) pi.

Table 1: System Identification Experiment 1 Results

| Model Operator | Avg SNR (dB) | Best SNR (dB) |
|---|---|---|
| shift | +2.7 | +3.6 |
| delta | -7.1 | +7.7 |
| gamma | +5.7 | +14.1 |
| rho | +9.7 | +16.5 |
| pi | +10.0 | +16.5 |

For the purposes of this experiment, we conducted several trials and selected $\theta(0)$ values which provided stable convergence. The values chosen for this experiment were: $\theta(0) = \{0.75, [0.5, 0.75], [0.75, 0.7, 0.35, -0.25]\}$ for the gamma, rho and pi operator models respectively. In each case we used model order $M = 8$.

Results for this experiment are shown in Table 1 and Figure 1. We observe that the pi operator gives the best performance overall. Some difficulties with instability occurring were encountered, thereby requiring a stability correction mechanism to be used on the operator updates. The next best performance was observed in the rho and then gamma models, with fewer instability problems occurring.

## 3.2  EXPERIMENT 2

The second experiment used a model described by

$$H(z) \;=\; \frac{1 - 0.8731z^{-1} - 0.8731z^{-2} + z^{-3}}{1 - 2.8653z^{-1} + 2.7505z^{-2} - 0.8843z^{-3}} \qquad (21)$$

This system is a 3rd order lowpass filter tested in [11]. The same experimental procedures as used in Experiment 1 were followed in this case.

For the second experiment (see Table 2), it was found that the pi operator gave the best results

Table 2: System Identification Experiment 2 Results

| Model Operator | Avg SNR (dB) | Best SNR (dB) |
|---|---|---|
| shift | 10.7 | 12.3 |
| delta | -21.5 | 10.2 |
| gamma | 13.5 | 15.0 |
| rho | 13.3 | 17.4 |
| pi | 14.0 | 17.9 |

recorded over all the tests. On average however, the improvement for this identification problem is less. It is observed that that the pi model is only slightly better than the gamma and rho models. Interestingly, the gamma and rho models had no problems with stability, while the pi model still suffered from convergence problems due to instability. As before, the delta model gave a wide variation in results and performed poorly.

From these and other experiments performed it appears that performance advantages can be obtained through the use of the more complex operators. As observed from the best recorded runs, the extra degrees of freedom in the rho and pi operators appear to provide the means to give better performance than the gamma model. The improvements of the more complex operators come at the expense of potential convergence problems due to instabilities occurring in the operators and a potentially multimodal mean square output error surface in the operator parameter space.

Clearly, there is a need for further investigation into the performance of these models on a wider range of tasks. We present these preliminary examples as an indication of how these alternative operators perform on some system identification problems.

# 4   CONCLUSIONS

Models based on the delta operator, rho operator, and pi operator have been presented and new algorithms derived. Comparisons have been made to the previously presented gamma model introduced by de Vries, Principe et. al. [4] for nonlinear signal processing applications.

While the simulation examples considered show are only linear, it is important to realize that the derivations are applicable for multilayer perceptrons, and that the input stage of these networks is identical to what we have considered here. We treat only the linear case in the examples in order not to complicate our understanding of the results, knowing that what happens in the input layer is important to higher layers in network structures.

The results obtained indicate that the more complex operators provide a potentially more powerful modelling structure, though there is a need for further work into mechanisms of maintaining stability while retaining good convergence properties.

The rho model was able to perform better than the gamma model on the problems tested, and gave similar results in terms of susceptibility to convergence and instability problems. The pi model appears capable of giving the best performance overall, but requires more attention to ensure the stability of the coefficients.

For future work it would be of value to analyse the convergence of the algorithms, in order to design methods which ensure stability can be maintained, while not disrupting the convergence of the model.

**Acknowledgements**

The first author acknowledges financial support from the Australian Research Council. The second author acknowledges partial support from the Australian Research Council.

**References**

[1] R.C. Agarwal and C.S. Burrus, "New recursive digital filter structures having very low sensitivity and roundoff noise", IEEE Trans. Circuits and Systems, vol. cas-22, pp. 921-927, Dec. 1975.

[2] de Vries, B. Principe, J.C. "A theory for neural networks with time delays", Advances in Neural Information Processing Systems, 3, R.P. Lippmann (Ed.), pp 162 - 168, 1991.

[3] de Vries, B., Principe, J. and P.G. de Oliveira "Adaline with adaptive recursive memory", Neural Networks for Signal Processing I. Juang, B.H., Kung, S.Y., Kamm, C.A. (Eds) IEEE Press, pp. 101-110, 1991.

[4] de Vries, B. Principe, J. "The Gamma Model – a new neural model for temporal processing". Neural Networks. Vol 5, No 4, pp 565 - 576, 1992.

[5] H. Fan and Q. Li, "A $\delta$ operator recursive gradient algorithm for adaptive signal processing", Proc. IEEE Int. Conf. Acoust. Speech and Signal Proc., vol. III, pp. 492-495, 1993.

[6] L. Ljung, and T. Söderström, Theory and Practice of Recursive Identification, Cambridge, Massachusetts: The MIT Press, 1983.

[7] R.H. Middleton, and G.C. Goodwin, Digital Control and Estimation, Englewood Cliffs: Prentice Hall, 1990.

[8] V. Peterka, "Control of Uncertain Processes: Applied Theory and Algorithms", Kybernetika, vol. 22, pp. 1-102, 1986.

[9] M. Palaniswami, "A new discrete-time operator for digital estimation and control". The University of Melbourne, Department of Electrical Engineering, Technical Report No.1, 1989.

[10] M. Palaniswami, "Digital Estimation and Control with a New Discrete Time Operator", Proc. 30th IEEE Conf. Decision and Control, pp. 1631-1632, 1991.

[11] J.C. Principe, B. de Vries, J-M. Kuo and P. Guedes de Oliveira, "Modeling Applications with the Focused Gamma Net", Advances in Neural Information Processing Systems, vol. 4, pp. 143-150, 1991.

[12] J.C. Principe, B. de Vries, and P. Guedes de Oliveira, "The Gamma Filter - a new class of adaptive IIR filters with restricted feedback", IEEE Trans. Signal Processing, vol. 41, pp. 649-656, 1993.

[13] G.V. Puskorius, and L.A. Feldkamp, "Decoupled Extended Kalman Filter Training of Feedforward Layered Networks", Proc. Int Joint Conf. Neural Networks, Seattle, vol I, pp. 771-777, 1991.

[14] E.B. Saff and A.D. Snider, Fundamentals of Complex Analysis for Mathematics, Science and Engineering. Englewood Cliffs, NJ: Prentice-Hall, 1976.

[15] S. Shah and F. Palmieri, "MEKA - A Fast Local Algorithm for Training Feedfoward Neural Networks", Proc Int Joint Conf. on Neural Networks, vol III, pp. 41-46, 1990.

[16] J.J. Shynk, "Adaptive IIR filtering using parallel-form realizations", IEEE Trans. Acoust. Speech Signal Proc., vol. 37, pp. 519-533, 1989.

[17] Soderstrom and Stoica, "System Identification", London: Prentice Hall, 1989.

[18] T.O. de Silva, P.G. de Oliveira, J.C. Principe, and B. de Vries, " Generalized feedforward filters with complex poles", Neural Networks for Signal Processing II, S.Y. Kung et. al. (Eds) Piscataway,NJ: IEEE Press, 1992.

[19] D. Williamson, "Delay replacement in direct form structures", IEEE Trans. Acoust., Speech, Signal Processing, vol. ASSP-34, pp. 453-460, Aprl. 1988.
